# Dual Kalman Filtering Methods for Nonlinear Prediction, Smoothing, and Estimation

**Eric A. Wan**
ericwan@ee.ogi.edu

**Alex T. Nelson**
atnelson@ee.ogi.edu

Department of Electrical Engineering
Oregon Graduate Institute
P.O. Box 91000 Portland, OR 97291

## Abstract

Prediction, estimation, and smoothing are fundamental to signal processing. To perform these interrelated tasks given noisy data, we form a time series model of the process that generates the data. Taking noise in the system explicitly into account, maximum-likelihood and Kalman frameworks are discussed which involve the dual process of estimating both the model parameters and the underlying state of the system. We review several established methods in the linear case, and propose several extensions utilizing dual Kalman filters (DKF) and forward-backward (FB) filters that are applicable to neural networks. Methods are compared on several simulations of noisy time series. We also include an example of nonlinear noise reduction in speech.

## 1  INTRODUCTION

Consider the general autoregressive model of a noisy time series with both process and additive observation noise:

$$x(k) = f(x(k-1), ...x(k-M), \mathbf{w}) + v(k-1) \tag{1}$$
$$y(k) = x(k) + r(k), \tag{2}$$

where $x(k)$ corresponds to the true underlying time series driven by process noise $v(k)$, and $f(\cdot)$ is a nonlinear function of past values of $x(k)$ parameterized by $\mathbf{w}$.

The only available observation is $y(k)$ which contains additional additive noise $r(k)$. *Prediction* refers to estimating an $\hat{x}(k)$ given past observations. (For purposes of this paper we will restrict ourselves to univariate time series.) In *estimation*, $\hat{x}(k)$ is determined given observations up to and including time $k$. Finally, *smoothing* refers to estimating $\hat{x}(k)$ given all observations, past and future.

The minimum mean square nonlinear prediction of $x(k)$ (or of $y(k)$) can be written as the conditional expectation $E[x(k)|\mathbf{x}(k-1)]$, where $\mathbf{x}(k) = [x(k), x(k-1), \cdots x(0)]$. If the time series $x(k)$ were directly available, we could use this data to generate an approximation of the optimal predictor. However, when $x(k)$ is not available (as is generally the case), the common approach is to use the noisy data directly, leading to an approximation of $E[y(k)|\mathbf{y}(k-1)]$. However, this results in a biased predictor: $E[y(k)|\mathbf{y}(k-1)] = E[x(k)|\mathbf{x}(k-1) + R(k-1)] \neq E[x(k)|\mathbf{x}(k-1)]$.

We may reduce the above bias in the predictor by exploiting the knowledge that the observations $y(k)$ are measurements arising from a time series. Estimates $\hat{x}(k)$ are found (either through *estimation* or *smoothing*) such that $||x(k) - \hat{x}(k)|| < ||x(k) - y(k)||$. These estimates are then used to form a predictor that approximates $E[x(k)|\hat{\mathbf{x}}(k-1)]$.[1]

In the remainder of this paper, we will develop methods for the dual estimation of both states $\hat{x}$ and weights $\hat{w}$. We show how a maximum-likelihood framework can be used to relate several existing algorithms and how established linear methods can be extended to a nonlinear framework. New methods involving the use of dual Kalman filters are also proposed and experiments are provided to compare results.

## 2   DUAL ESTIMATION

Given only noisy observations $y(k)$, the dual estimation problem requires consideration of both the standard prediction (or output) errors $e_p(k) = y(k) - f(\hat{\mathbf{x}}(k-1), \mathbf{w})$ as well as the observation (or input) errors $e_o(k) = y(k) - \hat{x}(k)$. The minimum observation error variance equals the noise variance $\sigma_r^2$. The prediction error, however, is correlated with the observation error since $y(k) - f(\mathbf{x}(k-1)) = r(k-1) + v(k)$, and thus has a minimum variance of $\sigma_r^2 + \sigma_v^2$. Assuming the errors are Gaussian, we may construct a log-likelihood function which is proportional to $\mathbf{e}^T \Sigma^{-1} \mathbf{e}$, where $\mathbf{e}^T = [e_o(0), e_o(1)....e_o(N), e_p(M), e_p(M+1), ...e_p(N)]$, a vector of all errors up to time N, and

$$\Sigma \triangleq E[\mathbf{e}\mathbf{e}^T] = \left[\begin{array}{cc|cc} & & 0 & 0 \\ & \sigma_r^2 I_N & \sigma_r^2 I_{N-M} & 0 \\ \hline 0 & \sigma_r^2 I_{N-M} & (\sigma_r^2 + \sigma_v^2) I_{N-M} \\ 0 & 0 & \end{array}\right] \tag{3}$$

Minimization of the log-likelihood function leads to the maximum-likelihood estimates for both $\hat{x}(k)$ and $\mathbf{w}$. (Although we may also estimate the noise variances $\sigma_r^2$ and $\sigma_v^2$, we will assume in this paper that they are known.) Two general frameworks for optimization are available:

## 2.1    Errors-In-Variables (EIV) Methods

This method comes from the statistics literature for nonlinear regression (see Seber and Wild, 1989), and involves batch optimization of the cost function in Equation 3. Only minor modifications are made to account for the time series model. These methods, however, are memory intensive ($\Sigma$ is approx. $2N \times 2N$) and also do not accommodate new data in an efficient manner. Retraining is necessary on all the data in order to produce estimates for the new data points.

If we ignore the cross correlation between the prediction and observation error, then $\Sigma$ becomes a diagonal matrix and the cost function may be expressed as simply $\sum_{k=1}^{N} \gamma e_p^2(k) + e_o^2(k)$, with $\gamma = \sigma_r^2/(\sigma_r^2 + \sigma_v^2)$. This is equivalent to the *Clearning* (CLRN) cost function (Weigend, 1995), developed as a heuristic method for cleaning the inputs in neural network modelling problems. While this allows for stochastic optimization, the assumption in the time series formulation may lead to severely biased results. Note also that no estimate is provided for the last point $\hat{x}(N)$.

When the model $f = \mathbf{w}^T \mathbf{x}$ is known and linear, EIV reduces to a standard (batch) weighted least squares procedure which can be solved in closed form to generate a maximum-likelihood estimate of the noise free time series. However, when the linear model is unknown, the problem is far more complicated. The inner product of the parameter vector $\mathbf{w}$ with the vector $\mathbf{x}(k-1)$ indicates a bilinear relationship between these unknown quantities. Solving for $x(k)$ requires knowledge of $\mathbf{w}$, while solving for $\mathbf{w}$ requires $x(k)$. Iterative methods are necessary to solve the nonlinear optimization, and a Newton's-type batch method is typically employed. An EIV method for nonlinear models is also readily developed, but the computational expense makes it less practical in the context of neural networks.

## 2.2    Kalman Methods

Kalman methods involve reformulation of the problem into a state-space framework in order to efficiently optimize the cost function in a recursive manner. At each time point, an optimal estimation is achieved by combining both a prior prediction and new observation. Connor (1994), proposed using an Extended Kalman filter with a neural network to perform state estimation alone. Puskorious and Feldkamp (1994) and others have posed the weight estimation in a state-space framework to allow Kalman training of a neural network. Here we extend these ideas to include the dual Kalman estimation of both states and weights for efficient maximum-likelihood optimization. We also introduce the use of forward-backward *information* filters and further explicate relationships to the EIV methods.

A state-space formulation of Equations 1 and 2 is as follows:

$$\mathbf{x}(k) = F[\mathbf{x}(k-1)] + Bv(k-1) \tag{4}$$
$$y(k) = C\mathbf{x}(k) + r(k) \tag{5}$$

where

$$\mathbf{x}(k) = \begin{bmatrix} x(k) \\ x(k-1) \\ \vdots \\ x(k-M+1) \end{bmatrix} \quad F[\mathbf{x}(k)] = \begin{bmatrix} f(x(k), \dots, x(k-M+1), \mathbf{w}) \\ x(k) \\ \vdots \\ x(k-M+2) \end{bmatrix} \quad B = \begin{bmatrix} 1 \\ 0 \\ \vdots \\ 0 \end{bmatrix}, \tag{6}$$

and $C = B^T$. If the model is linear, then $f(\mathbf{x}(k))$ takes the form $\mathbf{w}^T\mathbf{x}(k)$, and $F[\mathbf{x}(k)]$ can be written as $A\mathbf{x}(k)$, where $A$ is in controllable canonical form.

If the model is linear, and the parameters $\mathbf{w}$ are known, the Kalman filter (KF) algorithm can be readily used to estimate the states (see Lewis, 1986). At each time step, the filter computes the linear least squares estimate $\hat{x}(k)$ and prediction $\hat{x}^-(k)$, as well as their error covariances, $P_\mathbf{x}(k)$ and $P_\mathbf{x}^-(k)$. In the linear case with Gaussian statistics, the estimates are the minimum mean square estimates. With no prior information on $x$, they reduce to the maximum-likelihood estimates.

Note, however, that while the Kalman filter provides the maximum-likelihood estimate at each instant in time given all *past* data, the EIV approach is a batch method that gives a *smoothed* estimate given *all* data. Hence, only the estimates $\hat{x}(N)$ at the final time step will match. An exact equivalence for all time is achieved by combining the Kalman filter with a backwards *information filter* to produce a forward-backward (FB) smoothing filter (Lewis, 1986).[2] Effectively, an inverse covariance is propagated backwards in time to form backwards state estimates that are combined with the forward estimates. When the data set is large, the FB filter offers significant computational advantages over the batch form.

When the model is nonlinear, the Kalman filter cannot be applied directly, but requires a linearization of the nonlinear model at the each time step. The resulting algorithm is known as the extended Kalman filter (EKF) and effectively approximates the nonlinear function with a time-varying linear one.

### 2.2.1 Batch Iteration for Unknown Models

Again, when the linear model is unknown, the bilinear relationship between the time series estimates, $\hat{x}$, and the weight estimates, $\hat{\mathbf{w}}$ requires an iterative optimization. One approach (referred to as LS-KF) is to use a Kalman filter to estimate $\hat{x}(k)$ with $\hat{\mathbf{w}}$ fixed, followed by least-squares optimization to find $\hat{\mathbf{w}}$ using the current $\hat{x}(k)$. Specifically, the parameters are estimated as $\hat{\mathbf{w}} = (\mathbf{X}_{KF}^T\mathbf{X}_{KF})^{-1}\mathbf{X}_{KF}\mathbf{y}$, where $\mathbf{X}_{KF}$ is a matrix of KF state estimates, and $\mathbf{y}$ is a $1 \times N$ vector of observations.

For nonlinear models, we use a feedforward neural network to approximate $f(\cdot)$, and replace the LS and KF procedures by backpropagation and extended Kalman filtering, respectively (referred to here as BP-EKF, see Connor 1994). A disadvantage of this approach is slow convergence, due to keeping a set of *inaccurate* estimates fixed at each batch optimization stage.

### 2.2.2 Dual Kalman Filter

Another approach for unknown models is to concatenate both $\mathbf{w}$ and $\mathbf{x}$ into a joint state vector. The model and time series are then estimated simultaneously by applying an EKF to the nonlinear joint state equations (see Goodwin and Sin, 1994 for the linear case). This algorithm, however, has been known to have convergence problems.

An alternative is to construct a separate state-space formulation for the underlying weights as follows:

$$\mathbf{w}(k) = \mathbf{w}(k-1) \tag{7}$$

$$y(k) = f(\hat{\mathbf{x}}(k-1), \mathbf{w}(k)) + n(k), \tag{8}$$

where the state transition is simply an identity matrix, and $f(\hat{\mathbf{x}}(k-1), \mathbf{w}(k))$ plays the role of a time-varying nonlinear observation on $\mathbf{w}$.

When the unknown model is linear, the observation takes the form $\hat{\mathbf{x}}(k-1)^T \mathbf{w}(k)$. Then a pair of dual Kalman filters (DKF) can be run in parallel, one for state estimation, and one for weight estimation (see Nelson, 1976). At each time step, all current estimates are used. The dual approach essentially allows us to separate the non-linear optimization into two linear ones. Assumptions are that $\hat{\mathbf{x}}$ and $\hat{\mathbf{w}}$ remain uncorrelated and that statistics remain Gaussian. Note, however, that the error in each filter should be accounted for by the other. We have developed several approaches to address this coupling, but only present one here for the sake of brevity. In short, we write the variance of the noise $n(k)$ as $CP_{\mathbf{x}}^-(k)C^T + \sigma_r^2$. in Equation 8, and replace $v(k-1)$ by $v(k-1) + (\mathbf{w}(k)^T - \hat{\mathbf{w}}^T(k))\mathbf{x}(k-1)$ in Equation 4 for estimation of $\hat{\mathbf{x}}(k)$. Note that the ability to couple statistics in this manner is not possible in the batch approaches.

We further extend the DKF method to nonlinear neural network models by introducing a dual extended Kalman filtering method (DEKF). This simply requires that Jacobians of the neural network be computed for both filters at each time step. Note, by feeding $\hat{\mathbf{x}}(k)$ into the network, we are implicitly using a recurrent network.

### 2.2.3 Forward-Backward Methods

All of the Kalman methods can be reformulated by using forward-backward (FB) Kalman filtering to further improve state smoothing. However, the dual Kalman methods require an interleaving of the forward and backward state estimates in order to generate a smooth update at each time step. In addition, using the FB estimates requires caution because their noncausal nature can lead to a biased $\hat{\mathbf{w}}$ if they are used improperly. Specifically, for LS-FB the weights are computed as: $\hat{\mathbf{w}} = (\mathbf{X}_{\mathrm{KF}}^T \mathbf{X}_{\mathrm{FB}})^{-1} \mathbf{X}_{\mathrm{KF}} \mathbf{y}$ ,where $\mathbf{X}_{\mathrm{FB}}$ is a matrix of FB (smooth) state estimates. Equivalent adjustments are made to the dual Kalman methods. Furthermore, a model of the time-reversed system is required for the nonlinear case. The explication and results of these algorithms will be appear in a future publication.

## 3  EXPERIMENTS

Table 1 compares the different approaches on two linear time series, both when the linear model is known and when it is unknown. The least square (LS) estimation for the weights in the bottom row represents a baseline performance wherein no noise model is used. In-sample training set predictions must be interpreted carefully as all training set data is being used to optimize for the weights. We see that the Kalman-based methods perform better out of training set (recall the model-mismatch issue[1]). Further, only the Kalman methods allow for on-line estimations (on the test set, the state-estimation Kalman filters continue to operate with the weight estimates fixed). The forward-backward method further improves performance over KF methods. Meanwhile, the clearning-equivalent cost function sacrifices both state and weight estimation MSE for improved in-sample prediction; the resulting test set performance is significantly worse.

Several time series were used to compare the nonlinear methods, with the results summarized in Table 2. Conclusions parallel those for the linear case. Note, the DEKF method performed better than the baseline provided by standard backprop-

Table 1: Comparison of methods for two linear models
Model Known

|  | Train 1 | | Test 1 | | | Train 2 | | Test 2 | | |
|---|---|---|---|---|---|---|---|---|---|---|
|  | Est. | Pred. | Est. | Pred. | w | Est. | Pred. | Est. | Pred. | w |
| MLE | .094 | .322 | - | 1.09 | - | .165 | .558 | - | 1.32 | - |
| CLRN | .203 | .134 | - | 1.08 | - | .343 | .342 | - | 1.32 | - |
| KF | .134 | .559 | .132 | 0.59 | - | .197 | .778 | .221 | 0.85 | - |
| FB | .094 | .559 | .132 | 0.59 | - | .165 | .778 | .221 | 0.85 | - |

Model Unknown

|  | Est. | Pred. | Est. | Pred. | w | Est. | Pred. | Est. | Pred. | w |
|---|---|---|---|---|---|---|---|---|---|---|
| EIV |  |  |  |  |  | .172 | .545 | - | 1.81 | .122 |
| CLRN |  |  |  |  |  | .278 | .049 | - | 14.1 | 11.28 |
| LS-KF | .138 | .563 | .139 | .605 | .134 | .197 | .778 | .226 | 0.85 | .325 |
| LS-FB | .099 | .347 | .136 | .603 | .281 | .169 | .612 | .229 | 0.89 | .369 |
| DKF | .135 | .557 | .133 | .595 | .212 | .198 | .779 | .221 | .863 | .149 |
| DFB | .096 | .329 | .134 | .596 | .187 | .165 | .587 | .221 | .859 | .065 |
| LS | - | .886 | - | 1.09 | .612 | - | 1.08 | - | 1.32 | 0.590 |

MSE values for estimation (Est.), prediction (Pred.) and weights (w) (normalized to signal var.). 1 - AR(11) model, $\sigma_r^2 = 4$, $\sigma_v^2 = 1$, 2000 training samples, 1000 testing samples. EIV and CLRN were not computed for the unknown model due to memory constraints. 2 - AR(5) model, $\sigma_r^2 = .7.$, $\sigma_v^2 = .5.$, 375 training, 125 testing.

Table 2: Comparison of methods on nonlinear time series

|  | NNet 1 | | | | NNet 2 | | | | NNet 3 | | | |
|---|---|---|---|---|---|---|---|---|---|---|---|---|
|  | Train | | Test | | Train | | Test | | Train | | Test | |
|  | Es. | Pr. | Es. | Pr. | Es. | Pr. | Es. | Pr. | Es. | Pr. | Es. | Pr. |
| BP-EKF | .17 | .58 | .15 | .63 | .08 | .31 | .08 | .33 | .16 | .59 | .17 | .59 |
| DEKF | .14 | .57 | .13 | .59 | .07 | .30 | .06 | .32 | .14 | .56 | .14 | .55 |
| BP | *.35* | .57 | *.35* | .69 | *.22* | .30 | *.23* | .36 | *.32* | .68 | *.32* | .68 |

The series Nnet 1,2,3 are generated by autoregressive neural networks which exhibit limit cycle and chaotic behavior. $\sigma_v^2 = .16$, $\sigma_r^2 = .81$, 2700 training samples, 1300 testing samples. All network models fit using 10 inputs and 5 hidden units. Cross-validation was not used in any of the methods.

agation (wherein no model of the noise is used). The DEKF method exhibited fast convergence, requiring only 10-20 epochs for training. A DEFB method is under development.

The DEKF was tested on a speech signal corrupted with simulated bursting white noise (Figure 1). The method was applied to successive 64ms (512 point) windows of the signal, with a new window starting every 8ms (64 points). The results in the figure were computed assuming both $\sigma_v^2$ and $\sigma_r^2$ were known. The average SNR is improved by 9.94 dB. We also ran the experiment when $\sigma_r^2$ and $\sigma_v^2$ were estimated using only the noisy signal (Nelson and Wan, 1997), and acheived an SNR improvement of 8.50 dB. In comparison, available "state-of-the-art" techniques of *spectral subtraction* (Boll, 1979) and *RASTA* processing (Hermansky *et al.*, 1995), achieve SNR improvements of only .65 and 1.26 dB, respectively. We extend the algorithms to the colored noise case in a second paper (Nelson and Wan, 1997).

## 4   CONCLUSIONS

We have described various methods under a Kalman framework for the dual estimation of both states and weights of a noisy time series. These methods utilize both

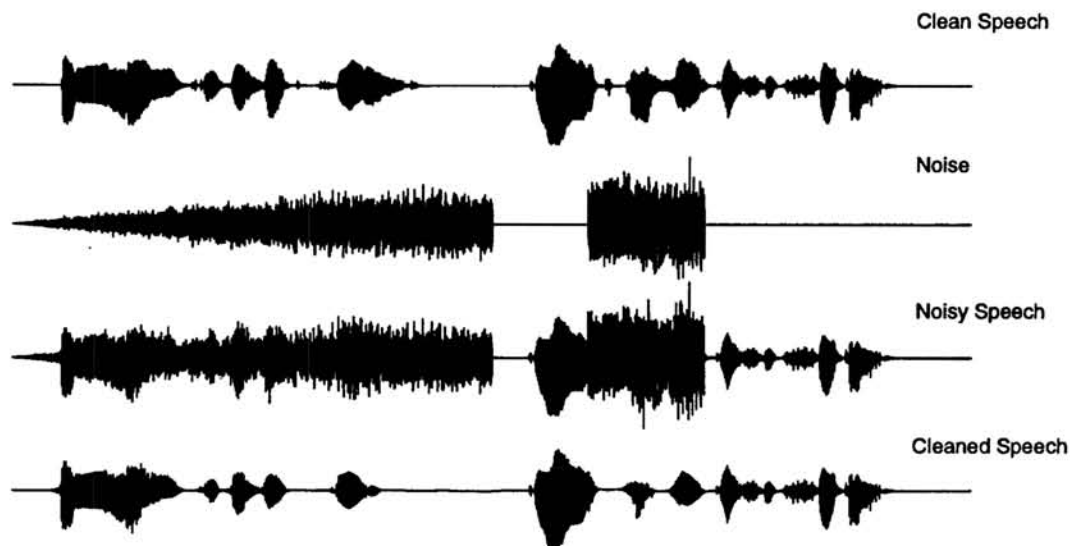

Figure 1: Cleaning Noisy Speech With The DEKF. 33,000 pts (5 sec.) shown.

process and observation noise models to improve estimation performance. Work in progress includes extensions for colored noise, blind signal separation, forward-backward filtering, and noise estimation. While further study is needed, the dual extended Kalman filter methods for neural network prediction, estimation, and smoothing offer potentially powerful new tools for signal processing applications.

## Acknowledgements

This work was sponsored in part by NSF under grant ECS-9410823 and by ARPA/AASERT Grant DAAH04-95-1-0485.

## Footnotes

[1] Because models are trained on estimated data $\hat{x}(k)$, it is important that estimated data still be used for prediction of out-of training set (on-line) data. In other words, if our model was formed as an approximation of $E[x(k)|\hat{\mathbf{x}}(k-1)]$, then we should not provide it with $y(k-1)$ as an input in order to avoid a model mismatch.

[2]A slight modification of the cost in Equation 3 is necessary to account for initial conditions in the Kalman form.

## References

S.F. Boll. Suppression of acoustic noise in speech using spectral subtraction. *IEEE ASSP-27*, pp. 113-120. April 1979.

J. Connor, R. Martin, L. Atlas. Recurrent neural networks and robust time series prediction. *IEEE Tr. on Neural Networks*. March 1994.

F. Lewis. *Optimal Estimation* John Wiley & Sons, Inc. New York. 1986.

G. Goodwin, K.S. Sin. *Adaptive Filtering Prediction and Control*. Prentice-Hall, Inc., Englewood Cliffs, NJ. 1994.

H. Hermansky, E. Wan, C. Avendano. Speech enhancement based on temporal processing. *ICASSP Proceedings*. 1995.

A. Nelson, E. Wan. Neural speech enhancement using dual extended Kalman filtering. Submitted to *ICNN'97*.

L. Nelson, E. Stear. The simultaneous on-line estimation of parameters and states in linear systems. *IEEE Tr. on Automatic Control*. February, 1976.

G. Puskorious, L. Feldkamp. Neural control of nonlinear dynamic systems with kalman filter trained recurrent networks. *IEEE Trn. on NN*, vol. 5, no. 2. 1994.

G. Seber, C. Wild. *Nonlinear Regression*. John Wiley & Sons. 1989.

A. Weigend, H.G. Zimmerman. Clearning. University of Colorado Computer Science Technical Report CU-CS-772-95. May, 1995.